# Impact of an Energy Normalization Transform on the Performance of the LF-ASD Brain Computer Interface

**Zhou Yu**[1]        **Steven G. Mason**[2]        **Gary E. Birch**[1,2]

[1] Dept. of Electrical and Computer Engineering
University of British Columbia
2356 Main Mall
Vancouver, B.C. Canada V6T 1Z4

[2] Neil Squire Foundation
220-2250 Boundary Road
Burnaby, B.C. Canada V5M 3Z3

## Abstract

**This paper presents an energy normalization transform as a method to reduce system errors in the LF-ASD brain-computer interface. The energy normalization transform has two major benefits to the system performance. First, it can increase class separation between the active and idle EEG data. Second, it can desensitize the system to the signal amplitude variability. For four subjects in the study, the benefits resulted in the performance improvement of the LF-ASD in the range from 7.7% to 18.9%, while for the fifth subject, who had the highest non-normalized accuracy of 90.5%, the performance did not change notably with normalization**.

## 1   Introduction

In an effort to provide alternative communication channels for people who suffer from severe loss of motor function, several researchers have worked over the past two decades to develop a direct Brain-Computer Interface (BCI). Since electroencephalographic (EEG) signal has good time resolution and is non-invasive, it is commonly used for data source of a BCI. A BCI system converts the input EEG into control signals, which are then used to control devices like computers, environmental control system and neuro-prostheses.

Mason and Birch [1] proposed the Low-Frequency Asynchronous Switch Design (LF-ASD) as a BCI which detected imagined voluntary movement-related potentials (IVMRPs) in spontaneous EEG. The principle signal processing components of the LF-ASD are shown in Figure 1.

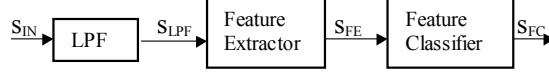

**Figure 1: The original LF-ASD design.**

The input to the low-pass filter (LPF), denoted as $S_{IN}$ in Figure 1, are six bipolar EEG signals recorded from $F_1$-$FC_1$, Fz-FCz, $F_2$-$FC_2$, $FC_1$-$C_1$, FCz-Cz and $FC_2$-$C_2$ sampled at 128 Hz. The cutoff frequency of the LPF implemented by Mason and Birch was 4 Hz. The Feature Extractor of the LF-ASD extracts custom features related to IVMRPs. The Feature Classifier implements a one-nearest-neighbor (1-NN) classifier, which determines if the input signals are related to a user state of voluntary movement or passive (idle) observation. The LF-ASD was able to achieve True Positive (TP) values in the range of 44%-81%, with the corresponding False Positive (FP) values around 1% [1].

Although encouraging, the current error rates of the LF-ASD are insufficient for real-world applications. This paper proposes a method to improve the system performance.

## 2 Design and Rationale

The improved design of the LF-ASD with the Energy Normalization Transform (ENT) is provided in Figure 2.

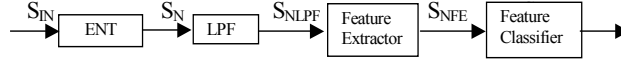

**Figure 2:** **The improved LF-ASD with the Energy Normalization Transform.**

The design of the Feature Extractor and Feature Classifier were the same as shown in Figure 1. The Energy Normalization Transform (ENT) is implemented as

$$S_N(n) = \frac{S_{IN}(n)}{\sqrt{\sum_{s=-(w_N-1)/2}^{s=(w_N-1)/2} S_{IN}^2(n-s) \Big/ w_N}}$$

where $W_N$ (normalization window size) is the only parameter in the equation. The optimal parameter value was obtained by exhaustive search for the best class separation between active and idle EEG data. The method of obtaining the active and idle EEG data is provided in Section 3.1.

The idea to use energy normalization to improve the LF-ASD design was based primarily on an observation that high frequency power decreases significantly around movement. For example, Jasper and Penfield [3] and Pfurtscheller et al, [4] reported EEG power decrease in the mu (8-12 Hz) and beta rhythm (18-26 Hz) when people are involved in motor related activity. Also Mason [5] found that the power in the frequency components greater than 4Hz decreased significantly during movement-related potential periods, while power in the frequency components less than 4Hz did not. Thus energy normalization, which would increase the low frequency power level, would strengthen the 0-4 Hz features used in the LF-ASD and hence reduce errors. In addition, as a side benefit, it can automatically adjust the mean scale of the input signal and desensitize the system to change in EEG power, which is known to vary over time [2]. Therefore, it was postulated that the addition of ENT into the improved design would have two major benefits. First, it can

increase the EEG power around motor potentials, consequently increasing the class separation and feature strength. Second, it can desensitize the system to amplitude variance of the input signal.

In addition, since the system components of the modified LF-ASD after the ENT were the same as in the original design, a major concern was whether or not the ENT distorted the features used by the LF-ASD. Since the features used by the LF-ASD are generated from the 0-4 Hz band, if the ENT does not distort the phase and magnitude spectrum in this specific band, it would not distort the features related to movement potential detection in the application.

# 3   Evaluation

## 3.1   Test data

Two types of EEG data were pre-recorded from five able-bodied individuals as shown in Figure 3.  Active Data Type and Idle Data Type. Active Data was recorded during repeated right index finger flexions alternating with periods of no motor activity; Idle Data was recorded during extended periods of passive observation.

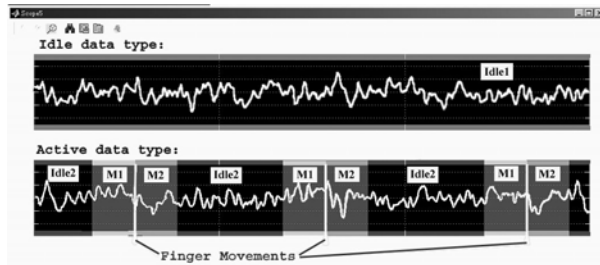

**Figure 3: Data Definition of M1, M2, Idle1 and Idle2.**

Observation windows centered at the time of the finger switch activations (as shown in Figure 4) were imposed in the active data to separate data related to movements from data during periods of idleness. For purpose of this study, data in the front part of the observation window was defined as M1 and data in the rear part of the window was defined as M2. Data falling out of the observation window was defined as Idle2.  All the data in the Idle Data Type was defined as Idle1 for comparison with Idle2.

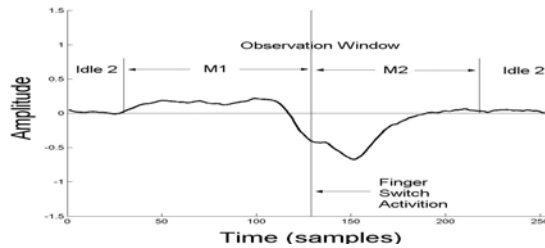

**Figure 4:  Ensemble Average of EEG centered on finger activations.**

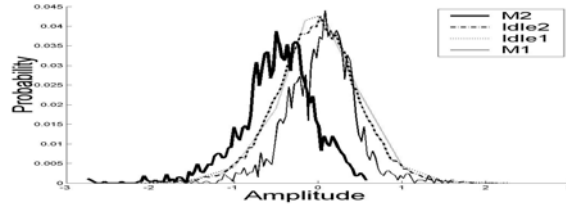

**Figure 5:  Density distribution of Idle1, Idle2, M1 and M2.**

It was noted, in terms of the density distribution of active and idle data, the separation between M2 and Idle2 was the largest and Idle1 and Idle2 were nearly identical (see Figure 5). For the study, M2 and Idle2 were chosen to represent the active and idle data classes and the separation between M2 and Idle2 data was defined by the difference of means (DOM) scaled by the amplitude range of Idle2.

### 3.2    Optimal parameter determination

The optimal combination of normalization window size, $W_N$, and observation window size, $W_O$ was selected to be that which achieved the maximal DOM value. This was determined by exhaustive search, and discussed in Section 4.1.

### 3.3    Effect of ENT on the Low Pass Filter output

As mentioned previously, it was postulated that the ENT had two major impacts: increasing the class separation between active and idle EEG and desensitizing the system to the signal amplitude variance. The hypothesis was evaluated by comparing characteristics of $S_{NLPF}$ and $S_{LPF}$ in Figure 1 and Figure 2. DOM was applied to measure the increased class separation. The signal with the larger DOM meant larger class separation. In addition, the signal with smaller standard deviation may result in a more stable feature set.

### 3.4    Effect of ENT on the LF-ASD output

The performances of the original and improved designs were evaluated by comparing the signal characteristics of $S_{NFC}$ in Figure 2 to $S_{FC}$ in Figure 1. A Receiver Operating Characteristic Curve (ROC Curve) [6] was generated for the original and improved designs. The ROC Curve characterizes the system performance over a range of TP vs. FP values. The larger area under ROC Curve indicates better system performance. In real applications, a BCI with high-level FP rates could cause frustration for subjects. Therefore, in this work only the LF-ASD performance when the FP values are less than 1% were studied.

## 4    Results

### 4.1    Optimal normalization window size ($W_N$)

The method to choose optimal $W_N$ was an exhaustive search for maximal DOM between active and idle classes. This method was possibly dependent on the observation window size ($W_O$). However, as shown in Figure 6a, the optimal $W_N$ was found to be independent of $W_O$. Experimentally, the $W_O$ values were selected in the range of 50-60 samples, which corresponded to largest DOM between non-normalized active and idle data. The optimal $W_N$ was obtained by exhaustive search

for the largest DOM through normalized active and idle data. The DOM vs. $W_N$ profile for Subject 1 is shown in Figure 6b.

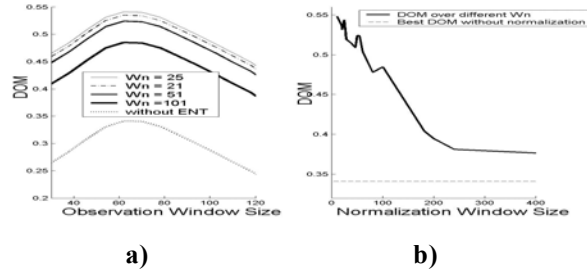

a)                                          b)

**Figure 6:  Optimal parameter determination for Subject 1 in Channel 1**
**a) DOM vs. $W_O$;  b) DOM vs. $W_N$.**

When using ENT, a small $W_N$ value may cause distortion to the feature set used by the LF-ASD. Thus, the optimal $W_N$ was not selected in this range (< 40 samples). When $W_N$ is greater than 200, the ENT has lost its capability to increase class separation and the DOM curve gradually goes towards the best separation without normalization. Thus, the optimal $W_N$ should correspond to the maximal DOM value when $W_N$ is in the range from 40 to 200. In Figure 6b, the optimal $W_N$ is around 51.

## 4.2    Effect of ENT on the Low Pass Filter output

With ENT, the standard deviation of the low frequency EEG signal decreased from around 1.90 to 1.30 over the six channels and over the five subjects. This change resulted in more stable feature sets. Thus, the ENT desensitizes the system to input signal variance.

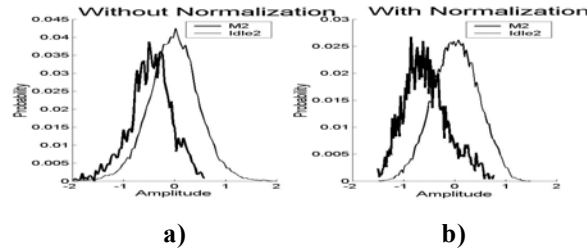

a)                                          b)

**Figure 7:  Density distribution of the active vs. idle class without**
**(a) and with (b) ENT, for Subject 1 in Channel 1.**

As shown in Figure 7, by increasing the EEG power around motor potentials, ENT can increase class separations between active and idle EEG data. The class separation in (frontal) Channels 1-3 across all subjects increased consistently with the proposed ENT. The same was true for (midline) Channels 4-6, for all subjects except Subject 5, whose DOM in channel 5-6 decreased by 2.3% and 3.4% respectively with normalization. That is consistent with the fact that his EEG power in Channels 4-6 does not decrease. On average, across all five subjects, DOM increases with normalization to about 28.8%, 26.4%, 39.4%, 20.5%, 17.8% and 22.5% over six channels respectively.

In addition, the magnitude and phase spectrums of the EEG signal before and after ENT is provided in Figure 8. The ENT has no visible distortion to the signal in the low frequency band (0-4 Hz) used by the LF-ASD. Therefore, the ENT does not distort the features used by the LF-ASD.

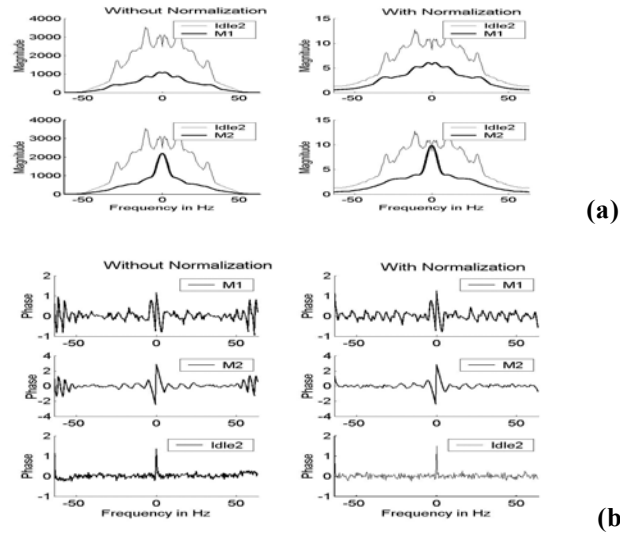

(a)

(b)

**Figure 8: Magnitude and phase spectrum of the EEG signal before and after ENT.**

## 4.3    Effect of ENT on the LF-ASD output

The two major benefits of the ENT to the low frequency EEG data result in the performance improvement of the LF-ASD.    Subject 1's ROC Curves with and without ENT is shown in Figure 9, where the ROC-Curve with ENT of optimal parameter value is above the ROC Curve without ENT. This indicates that the improved LF-ASD performs better. Table I compares the system performance with and without ENT in terms of TP with corresponding FP at 1% across all the 5 subjects.

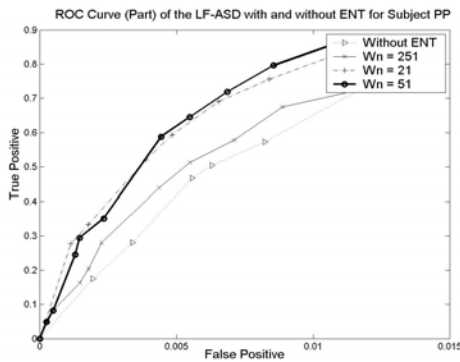

**Figure 9:  The ROC Curves (in the section of interest) of Subject 1 with different $W_N$ values and the corresponding ROC Curve without ENT.**

**Table I: Performance of the LF-ASD with and without LF-ASD in terms of the True Positive rate with corresponding False Positive at 1%.**

|  | TP without ENT | TP with ENT | Performance Improvement |
|---|---|---|---|
| Subject 1 | 66.1% | 85.0% | 18.9% |
| Subject 2 | 82.7% | 90.4% | 7.7% |
| Subject 3 | 79.7% | 88.0% | 8.3% |
| Subject 4 | 79.3% | 87.8% | 8.5% |
| Subject 5 | 90.5% | 88.7% | -1.8% |

For 4 out of 5 subjects, corresponding with the FP at 1%, the improved system with ENT increased the TP value by 7.7%, 8.3%, 8.5% and 18.9% respectively. Thus, for these subjects, the range of TP with FP at 1% was improved from 66.1%-82.7% to 85.0%-90.4% with ENT. For the fifth subject, who had the highest non-normalized accuracy of 90.5%, the performance remained around 90% with ENT. In addition, this evaluation is conservative. Since the codebook in the Feature Classifier and the parameters in the Feature Extractor of the LF-ASD were derived from non-normalized EEG, they work in favor of the non-normalized EEG. Therefore, if the parameters and the codebook of the modified LF-ASD are generated from the normalized EEG in the future, the modified LF-ASD may show better performance than this evaluation.

## 5   Conclusion

The evaluation with data from five able-bodied subjects indicates that the proposed system with Energy Normalization Transform (ENT) has better performance than the original. This study has verified the original hypotheses that the improved design with ENT might have two major benefits: increased the class separation between active and idle EEG and desensitized the system performance to input amplitude variance. As a side benefit, the ENT can also make the design less sensitive to the mean input scale.

In the broad band, the Energy Normalization Transform is a non-linear transform. However, it has no visible distortion to the signal in the 0-4 Hz band. Therefore, it does not distort the features used by the LF-ASD.

For 4 out of 5 subjects, with the corresponding False Positive rate at 1%, the proposed transform increased the system performance by 7.7%, 8.3%, 8.5% and 18.9% respectively in terms of True Positive rate. Thus, the overall performance of the LF-ASD for these subjects was improved from 66.1%-82.7% to 85.0%-90.4%. For the fifth subject, who had the highest non-normalized accuracy of 90.5%, the performance did not change notably with normalization. In the future with the codebook derived from the normalized data, the performance could be further improved.

## References

[1] Mason, S. G. and Birch, G. E., (2000) A Brain-Controlled Switch for Asynchronous Control Applications. *IEEE Trans Biomed Eng*, **47**(10):1297-1307.

[2] Vaughan, T. M., Wolpaw, J. R., and Donchin, E. (1996) EEG-Based Communication: Prospects and Problems. *IEEE Trans Reh Eng*, **4**(4):425-430.

[3] Jasper, H. and Penfield, W. (1949) Electrocortiograms in man: Effect of voluntary movement upon the electrical activity of the precentral gyrus. *Arch.Psychiat.Nervenkr.*, **183**:163-174.

[4] Pfurtscheller, G., Neuper, C., and Flotzinger, D. (1997) EEG-based discrimination between imagination of right and left hand movement. *Electroencephalography and Clinical Neurophysiology*, **103**:642-651.

[5] Mason, S. G. (1997) Detection of single trial index finger flexions from continuous, spatiotemporal EEG. *PhD Thesis*, UBC, January.

[6] Green, D. M. and Swets, J. A. (1996) *Signal Detection Theory and Psychophysics* New York: John Wiley and Sons, Inc.
